# Learning to Classify Galaxy Shapes Using the EM Algorithm

**Sergey Kirshner**
Information and Computer Science
University of California
Irvine, CA 92697-3425
skirshne@ics.uci.edu

**Igor V. Cadez**
Sparta Inc.,
23382 Mill Creek Drive #100,
Laguna Hills, CA 92653
igor_cadez@sparta.com

**Padhraic Smyth**
Information and Computer Science
University of California
Irvine, CA 92697-3425
smyth@ics.uci.edu

**Chandrika Kamath**
Center for Applied Scientific Computing
Lawrence Livermore National Laboratory
Livermore, CA 94551
kamath2@llnl.gov

## Abstract

We describe the application of probabilistic model-based learning to the problem of automatically identifying classes of galaxies, based on both morphological and pixel intensity characteristics. The EM algorithm can be used to learn how to spatially orient a set of galaxies so that they are geometrically aligned. We augment this "ordering-model" with a mixture model on objects, and demonstrate how classes of galaxies can be learned in an unsupervised manner using a two-level EM algorithm. The resulting models provide highly accurate classification of galaxies in cross-validation experiments.

## 1 Introduction and Background

The field of astronomy is increasingly data-driven as new observing instruments permit the rapid collection of massive archives of sky image data. In this paper we investigate the problem of identifying bent-double radio galaxies in the FIRST (Faint Images of the Radio Sky at Twenty-cm) Survey data set [1]. FIRST produces large numbers of radio images of the deep sky using the Very Large Array at the National Radio Astronomy Observatory. It is scheduled to cover more that 10,000 square degrees of the northern and southern caps (skies). Of particular scientific interest to astronomers is the identification and cataloging of sky objects with a "bent-double" morphology, indicating clusters of galaxies ([8], see Figure 1). Due to the very large number of observed deep-sky radio sources, (on the order of $10^6$ so far) it is infeasible for the astronomers to label all of them manually.

The data from the FIRST Survey (http://sundog.stsci.edu/) is available in both raw image format and in the form of a catalog of features that have been automatically derived from the raw images by an image analysis program [8]. Each entry corresponds to a single detectable "blob" of bright intensity relative to the sky background: these entries are called

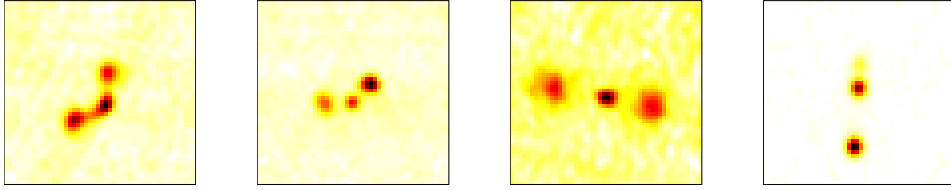

Figure 1: 4 examples of radio-source galaxy images. The two on the left are labelled as "bent-doubles" and the two on the right are not. The configurations on the left have more "bend" and symmetry than the two non-bent-doubles on the right.

**components**. The "blob" of intensities for each component is fitted with an ellipse. The ellipses and intensities for each component are described by a set of estimated features such as sky position of the centers (RA (right ascension) and Dec (declination)), peak density flux and integrated flux, root mean square noise in pixel intensities, lengths of the major and minor axes, and the position angle of the major axis of the ellipse counterclockwise from the north. The goal is to find sets of components that are spatially close and that resemble a bent-double. In the results in this paper we focus on candidate sets of components that have been detected by an existing spatial clustering algorithm [3] where each set consists of three components from the catalog (three ellipses). As of the year 2000, the catalog contained over 15,000 three-component configurations and over 600,000 configurations total. The set which we use to build and evaluate our models consists of a total of 128 examples of bent-double galaxies and 22 examples of non-bent-double configurations. A configuration is labelled as a bent-double if two out of three astronomers agree to label it as such. Note that the visual identification process is the bottleneck in the process since it requires significant time and effort from the scientists, and is subjective and error-prone, motivating the creation of automated methods for identifying bent-doubles.

Three-component bent-double configurations typically consist of a center or "core" component and two other side components called "lobes". Previous work on automated classification of three-component candidate sets has focused on the use of decision-tree classifiers using a variety of geometric and image intensity features [3]. One of the limitations of the decision-tree approach is its relative inflexibility in handling uncertainty about the object being classified, e.g., the identification of which of the three components should be treated as the core of a candidate object. A bigger limitation is the fixed size of the feature vector. A primary motivation for the development of a probabilistic approach is to provide a framework that can handle uncertainties in a flexible coherent manner.

## 2    Learning to Match Orderings using the EM Algorithm

We denote a three-component **configuration** by $\mathcal{C} = (\mathbf{c}_1, \mathbf{c}_2, \mathbf{c}_3)$, where the $\mathbf{c}_i$'s are the components (or "blobs") described in the previous section. Each component $\mathbf{c}_x$ is represented as a feature vector, where the specific features will be defined later. Our approach focuses on building a probabilistic model for bent-doubles: $p(\mathcal{C}) = p(\mathbf{c}_1, \mathbf{c}_2, \mathbf{c}_3)$, the likelihood of the observed $\mathbf{c}_i$ under a bent-double model where we implicitly condition (for now) on the class "bent-double."

By looking at examples of bent-double galaxies and by talking to the scientists studying them, we have been able to establish a number of potentially useful characteristics of the components, the primary one being geometric symmetry. In bent-doubles, two of the components will look close to being mirror images of one another with respect to a line through the third component. We will call mirror-image components *lobe* compo-

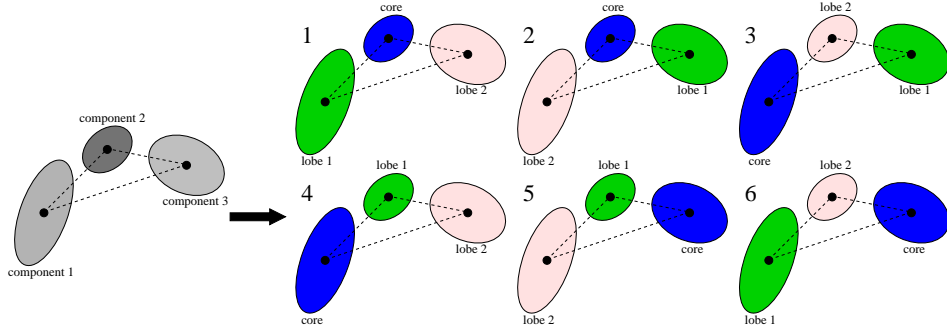

Figure 2: Possible orderings for a hypothetical bent-double. A good choice of ordering would be either 1 or 2.

nents, and the other one the *core* component. It also appears that non-bent-doubles either don't exhibit such symmetry, or the angle formed at the core component is too straight—the configuration is not "bent" enough. Once the core component is identified, we can calculate symmetry-based features. However, identifying the most plausible core component requires either an additional algorithm or human expertise. In our approach we use a probabilistic framework that averages over different possible orderings weighted by their probability given the data.

In order to define the features, we first need to determine the mapping of the components to labels "core", "lobe 1", and "lobe 2" ($c$, $l_1$, and $l_2$ for short). We will call such a mapping an **ordering**. Figure 2 shows an example of possible orderings for a configuration. We can number the orderings $1, \ldots, 6$. We can then write

$$p\left(\mathcal{C}\right) \;=\; \sum_{k=1}^{6} p\left(\mathbf{c}_c, \mathbf{c}_{l_1}, \mathbf{c}_{l_2} | \Omega = k\right) p\left(\Omega = k\right), \tag{1}$$

i.e., a mixture over all possible orientations. Each ordering is assumed a priori to be equally likely, i.e., $p(\Omega = k) = \frac{1}{6}$. Intuitively, for a configuration that clearly looks like a bent-double the terms in the mixture corresponding to the correct ordering would dominate, while the other orderings would have much lower probability.

We represent each component $\mathbf{c}_x$ by $M$ features (we used $M = 3$). Note that the features can only be calculated conditioned on a particular mapping since they rely on properties of the (assumed) core and lobe components. We denote by $f_{mk}\left(\mathcal{C}\right)$ the values corresponding to the $m$th feature for configuration $\mathcal{C}$ under the ordering $\Omega = k$, and by $f_{mkj}\left(\mathcal{C}\right)$ we denote the feature value of component $j$: $f_{mk}\left(\mathcal{C}\right) = \left(f_{mk1}\left(\mathcal{C}\right), \ldots, f_{mkB_m}\left(\mathcal{C}\right)\right)$ (in our case, $B_m = 3$ is the number of components). Conditioned on a particular mapping $\Omega = k$, where $x \in \{c, l_1, l_2\}$ and $c, l_1, l_2$ are defined in a cyclical order, our features are defined as:

- $f_{1k}\left(\mathcal{C}\right)$ : Log-transformed angle, the angle formed at the center of the component (a vertex of the configuration) mapped to label $x$;

- $f_{2k}\left(\mathcal{C}\right)$ : Logarithms of side ratios, $\dfrac{\left|\text{center of } x \text{ to center of } next(x)\right|}{\left|\text{center of } x \text{ to center of } prev(x)\right|}$;

- $f_{3k}\left(\mathcal{C}\right)$ : Logarithms of intensity ratios, $\dfrac{\text{peak flux of } next(x)}{\text{peak flux of } prev(x)}$,

and so $\left(\mathcal{C} | \Omega = k\right) = \left(f_{1k}\left(\mathcal{C}\right), f_{2k}\left(\mathcal{C}\right) f_{3k}\left(\mathcal{C}\right)\right)$ for a 9-dimensional feature vector in total. Other features are of course also possible. For our purposes in this paper this particular set appears to capture the more obvious visual properties of bent-double galaxies.

For a set $D = \{\mathbf{d}_1, \ldots, \mathbf{d}_N\}$ of configurations, under an i.i.d. assumption for configurations, we can write the likelihood as

$$P(D) = \prod_{i=1}^{N} \sum_{k=1}^{K} P(\Omega_i = k) P(f_{1k}(\mathbf{d}_i), \ldots, f_{Mk}(\mathbf{d}_i)),$$

where $\Omega_i$ is the ordering for configuration $\mathbf{d}_i$. While in the general case one can model $P(f_{1k}(\mathbf{d}_i), \ldots, f_{Mk}(\mathbf{d}_i))$ as a full joint distribution, for the results reported in this paper we make a number of simplifying assumptions, motivated by the fact that we have relatively little labelled training data available for model building. First, we assume that the $f_{mk}(\mathbf{d}_i)$ are conditionally independent. Second, we are also able to reduce the number of components for each $f_{mk}(\mathbf{d}_i)$ by noting functional dependencies. For example, given two angles of a triangle, we can uniquely determine the third one. We also assume that the remaining components for each feature are conditionally independent. Under these assumptions the multivariate joint distribution $P(f_{1k}(\mathbf{d}_i), \ldots, f_{Mk}(\mathbf{d}_i))$ is factored into a product of simple distributions, which (for the purposes of this paper) we model using Gaussians. If we know for every training example which component should be mapped to label $c$, we can then unambiguously estimate the parameters for each of these distributions.

In practice, however, the identity of the core component is unknown for each object. Thus, we use the EM algorithm to automatically estimate the parameters of the above model. We begin by randomly assigning an ordering to each object. For each subsequent iteration the E-step consists of estimating a probability distribution over possible orderings for each object, and the M-step estimates the parameters of the feature-distributions using the probabilistic ordering information from the E-step. In practice we have found that the algorithm converges relatively quickly (in 20 to 30 iterations) on both simulated and real data. It is somewhat surprising that this algorithm can reliably "learn" how to align a set of objects, without using any explicit objective function for alignment, but instead based on the fact that feature values for certain orderings exhibit a certain self-consistency relative to the model. Intuitively it is this self-consistency that leads to higher-likelihood solutions and that allows EM to effectively align the objects by maximizing the likelihood.

After the model has been estimated, the likelihood of new objects can also be calculated under the model, where the likelihood now averages over all possible orderings weighted by their probability given the observed features.

The problem described above is a specific instance of a more general *feature unscrambling* problem. In our case, we assume that configurations of three 3-dimensional components (i.e. 3 features) each are generated by some distribution. Once the objects are generated, the orders of their components are permuted or scrambled. The task is then to simultaneously learn the parameters of the original distributions and the scrambling for each object. In the more general form, each configuration consists of $L$ $M$-dimensional configurations. Since there are $L!$ possible orderings of $L$ components, the problem becomes computationally intractable if $L$ is large. One solution is to restrict the types of possible scrambles (to cyclic shifts for example).

## 3 Automatic Galaxy Classification

We used the algorithm described in the previous section to estimate the parameters of features and orderings of the bent-double class from labelled training data and then to rank candidate objects according to their likelihood under the model. We used leave-one-out cross-validation to test the classification ability of this supervised model, where for each of the 150 examples we build a model using the positive examples from the set of 149 "other" examples, and then score the "left-out" example with this model. The examples are then sorted in decreasing order by their likelihood score (averaging over different possi-

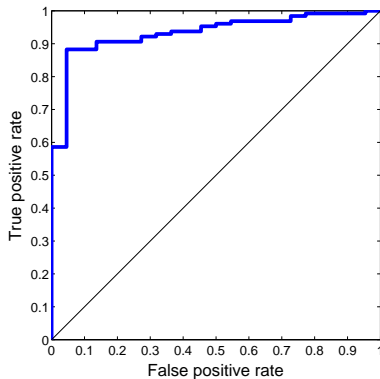

Figure 3: ROC plot for a model using angle, ratio of sides, and ratio of intensities, as features, and learned using ordering-EM with labelled data.

ble orderings) and the results are analyzed using a receiver operating characteristic (ROC) methodology. We use $A_{ROC}$, the area under the curve, as a measure of goodness of the model, where a perfect model would have $A_{ROC} = 1$ and random performance corresponds to $A_{ROC} = 0.5$. The supervised model, using EM for learning ordering models, has a cross-validated $A_{ROC}$ score of 0.9336 (Figure 3) and appears to be quite useful at detecting bent-double galaxies.

## 4  Model-Based Galaxy Clustering

A useful technique in understanding astronomical image data is to cluster image objects based on their morphological and intensity properties. For example, consider how one might cluster the image objects in Figure 1 into clusters, where we have features on angles, intensities, and so forth. Just as with classification, clustering of the objects is impeded by not knowing which of the "blobs" corresponds to the true "core" component.

From a probabilistic viewpoint, clustering can be treated as introducing another level of hidden variables, namely the unknown class (or cluster) identity of each object. We can generalize the EM algorithm for orderings (Section 2) to handle this additional hidden level. The model is now a mixture of clusters where each cluster is modelled as a mixture of orderings. This leads to a more complex two-level EM algorithm than that presented in Section 2, where at the inner-level the algorithm is learning how to orient the objects, and at the outer level the algorithm is learning how to group the objects into $C$ classes. Space does not permit a detailed presentation of this algorithm—however, the derivation is straightforward and produces intuitive update rules such as:

$$\hat{\mu}_{cmj} = \frac{1}{N\hat{P}\left(cl = c|\Theta\right)} \sum_{i=1}^{N} \sum_{k=1}^{K} P\left(cl_i = c|\Omega_i = k, D, \Theta\right) P\left(\Omega_i = k|D, \Theta\right) f_{mkj}\left(\mathbf{d}_i\right)$$

where $\mu_{cmj}$ is the mean for the $c$th cluster ($1 \leq c \leq C$), the $m$th feature ($1 \leq m \leq M$), and the $j$th component of $f_{mk}\left(\mathbf{d}_i\right)$, and $\Omega_i = k$ corresponds to ordering $k$ for the $i$th object.

We applied this algorithm to the data set of 150 sky objects, where unlike the results in Section 3, the algorithm now had no access to the class labels. We used the Gaussian conditional-independence model as before, and grouped the data into $K = 2$ clusters. Figures 4 and 5 show the highest likelihood objects, out of 150 total objects, under the

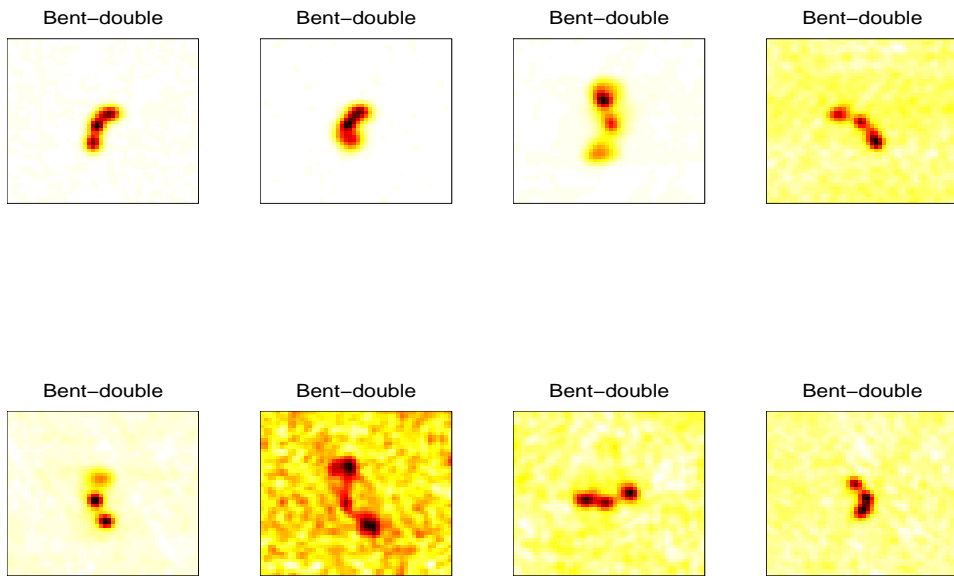

Figure 4: The 8 objects with the highest likelihood conditioned on the model for the **larger** of the two clusters learned by the unsupervised algorithm.

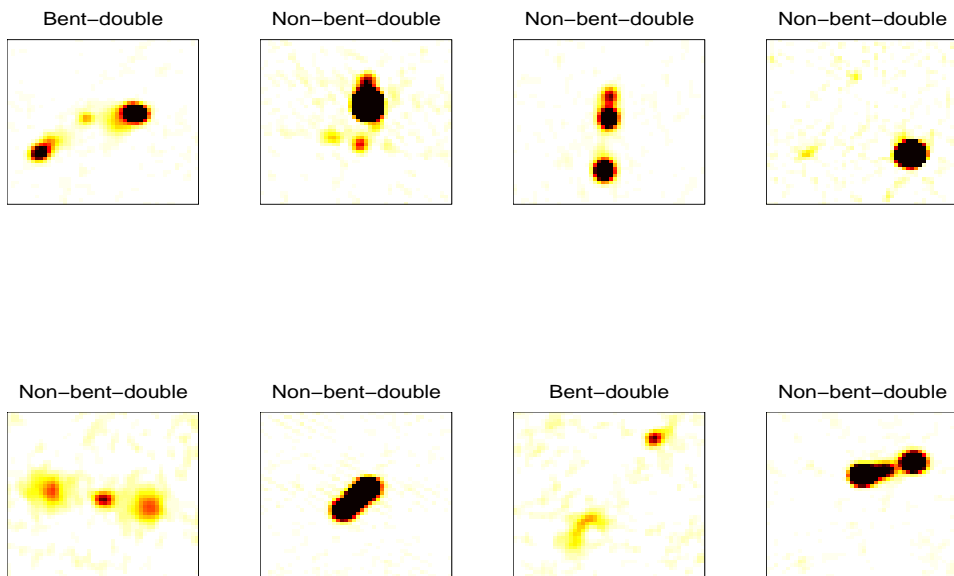

Figure 5: The 8 objects with the highest likelihood conditioned on the model for the **smaller** of the two clusters learned by the unsupervised algorithm.

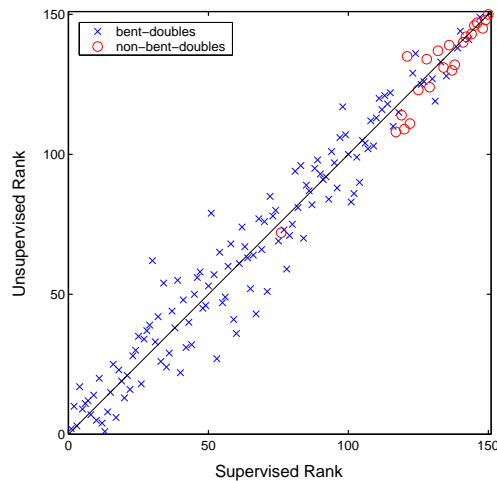

Figure 6: A scatter plot of the ranking from the unsupervised model versus that of the supervised model.

models for the larger cluster and smaller cluster respectively. The larger cluster is clearly a bent-double cluster: 89 of the 150 objects are more likely to belong to this cluster under the model and 88 out of the 89 objects in this cluster have the bent-double label. In other words, the unsupervised algorithm has discovered a cluster that corresponds to "strong examples" of bent-doubles relative to the particular feature-space and model. In fact the non-bent-double that is assigned to this group may well have been mislabelled (image not shown here). The objects in Figure 5 are clearly inconsistent with the general visual pattern of bent-doubles and this cluster consists of a mixture of non-bent-double and "weaker" bent-double galaxies. The objects in Figures 5 that are labelled as bent-doubles seem quite atypical compared to the bent-doubles in Figure 4.

A natural hypothesis is that cluster 1 (88 bent-doubles) in the unsupervised model is in fact very similar to the supervised model learned using the labelled set of 128 bent-doubles in Section 3. Indeed the parameters of the two Gaussian models agree quite closely and the similarity of the two models is illustrated clearly in Figure 6 where we plot the likelihood-based ranks of the unsupervised model versus those of the supervised model. Both models are in close agreement and both are clearly performing well in terms of separating the objects in terms of their class labels.

## 5   Related Work and Future Directions

A related earlier paper is Kirshner et al [6] where we presented a heuristic algorithm for solving the orientation problem for galaxies. The generalization to an EM framework in this paper is new, as is the two-level EM algorithm for clustering objects in an unsupervised manner.

There is a substantial body of work in computer vision on solving a variety of different object matching problems using probabilistic techniques—see Mjolsness [7] for early ideas and Chui et al. [2] for a recent application in medical imaging. Our work here differs in a number of respects. One important difference is that we use EM to learn a model for the simultaneous correspondence of $N$ objects, using both geometric and intensity-based features, whereas prior work in vision has primarily focused on matching one object to

another (essentially the $N = 2$ case). An exception is the recent work of Frey and Jojic [4, 5] who used a similar EM-based approach to simultaneously cluster images and estimate a variety of local spatial deformations. The work described in this paper can be viewed as an extension and application of this general methodology to a real-world problem in galaxy classification.

Earlier work on bent-double galaxy classification used decision tree classifiers based on a variety of geometric and intensity-based features [3]. In future work we plan to compare the performance of this decision tree approach with the probabilistic model-based approach proposed in this paper. The model-based approach has some inherent advantages over a decision-tree model for these types of problems. For example, it can directly handle objects in the catalog with only 2 blobs or with 4 or more blobs by integrating over missing intensities and over missing correspondence information using mixture models that allow for missing or extra "blobs". Being able to classify such configurations automatically is of significant interest to the astronomers.

## Acknowledgments

This work was performed under a sub-contract from the ASCI Scientific Data Management Project of the Lawrence Livermore National Laboratory. The work of S. Kirshner and P. Smyth was also supported by research grants from NSF (award IRI-9703120), the Jet Propulsion Laboratory, IBM Research, and Microsoft Research. I. Cadez was supported by a Microsoft Graduate Fellowship. The work of C. Kamath was performed under the auspices of the U.S. Department of Energy by University of California Lawrence Livermore National Laboratory under contract No. W-7405-Eng-48. We gratefully acknowledge our FIRST collaborators, in particular, Robert H. Becker for sharing his expertise on the subject.

## References

[1] R. H. Becker, R. L. White, and D. J. Helfand. The FIRST Survey: Faint Images of the Radio Sky at Twenty-cm. *Astrophysical Journal*, 450:559, 1995.

[2] H. Chui, L. Win, R. Schultz, J. S. Duncan, and A. Rangarajan. A unified feature registration method for brain mapping. In *Proceedings of Information Processing in Medical Imaging*, pages 300–314. Springer-Verlag, 2001.

[3] I. K. Fodor, E. Cantú-Paz, C. Kamath, and N. A. Tang. Finding bent-double radio galaxies: A case study in data mining. In *Proceedings of the Interface: Computer Science and Statistics Symposium*, volume 33, 2000.

[4] B. J. Frey and N. Jojic. Estimating mixture models of images and inferring spatial transformations using the EM algorithm. In *Proceedings of IEEE Computer Society Conference on Computer Vision and Pattern Recognition*, 1999.

[5] N. Jojic and B. J. Frey. Topographic transformation as a discrete latent variable. In *Advances in Neural Information Processing Systems 12*. MIT Press, 2000.

[6] S. Kirshner, I. V. Cadez, P. Smyth, C. Kamath, and E. Cantú-Paz. Probabilistic model-based detection of bent-double radio galaxies. In *Proceedings 16th International Conference on Pattern Recognition*, volume 2, pages 499–502, 2002.

[7] E. Mjolsness. Bayesian inference on visual grammars by neural networks that optimize. Technical Report YALEU/DCS/TR-854, Department of Computer Science, Yale University, May 1991.

[8] R. L. White, R. H. Becker, D. J. Helfand, and M. D. Gregg. A catalog of 1.4 GHz radio sources from the FIRST Survey. *Astrophysical Journal*, 475:479, 1997.
